# Cholinergic Modulation May Enhance Cortical Associative Memory Function

**Michael E. Hasselmo***
Computation and
Neural Systems
Caltech 216-76
Pasadena, CA 91125

**Brooke P. Anderson†**
Computation and
Neural Systems
Caltech 139-74
Pasadena, CA 91125

**James M. Bower**
Computation and
Neural Systems
Caltech 216-76
Pasadena, CA 91125

## Abstract

Combining neuropharmacological experiments with computational model-
ing, we have shown that cholinergic modulation may enhance associative
memory function in piriform (olfactory) cortex. We have shown that the
acetylcholine analogue carbachol selectively suppresses synaptic transmis-
sion between cells within piriform cortex, while leaving input connections
unaffected. When tested in a computational model of piriform cortex,
this selective suppression, applied during learning, enhances associative
memory performance.

## 1   INTRODUCTION

A wide range of behavioral studies support a role for the neurotransmitter acetyl-
choline in memory function (Kopelman, 1986; Hagan and Morris, 1989). However,
the role of acetylcholine in memory function has not been linked to the specific
neuropharmacological effects of this transmitter within cerebral cortical networks.
For several years, we have explored cerebral cortical associative memory function
using the piriform cortex as a model system (Wilson and Bower, 1988, Bower, 1990;
Hasselmo *et al.*, 1991). The anatomical structure of piriform cortex (represented
schematically in figure 1) shows the essential features of more abstract associative

*e-mail: hasselmo@smaug.cns.caltech.edu
†e-mail: brooke@hope.caltech.edu

matrix memory models (Haberly and Bower, 1989) [1]. Afferent fibers in layer 1a provide widely distributed input, while intrinsic fibers in layer 1b provide extensive excitatory connections between cells within the cortex. Computational models of piriform cortex demonstrate a theoretical capacity for associative memory function (Wilson and Bower, 1988; Bower, 1990; Hasselmo *et al.*, 1991). Recently, we have investigated differences in the physiological properties of the afferent and intrinsic fiber systems, using modeling to test how these differences affect memory function. In the experiments described below, we found a selective cholinergic suppression of intrinsic fiber synaptic transmission. When tested in a simplified model of piriform cortex, this modulation enhances associative memory performance.

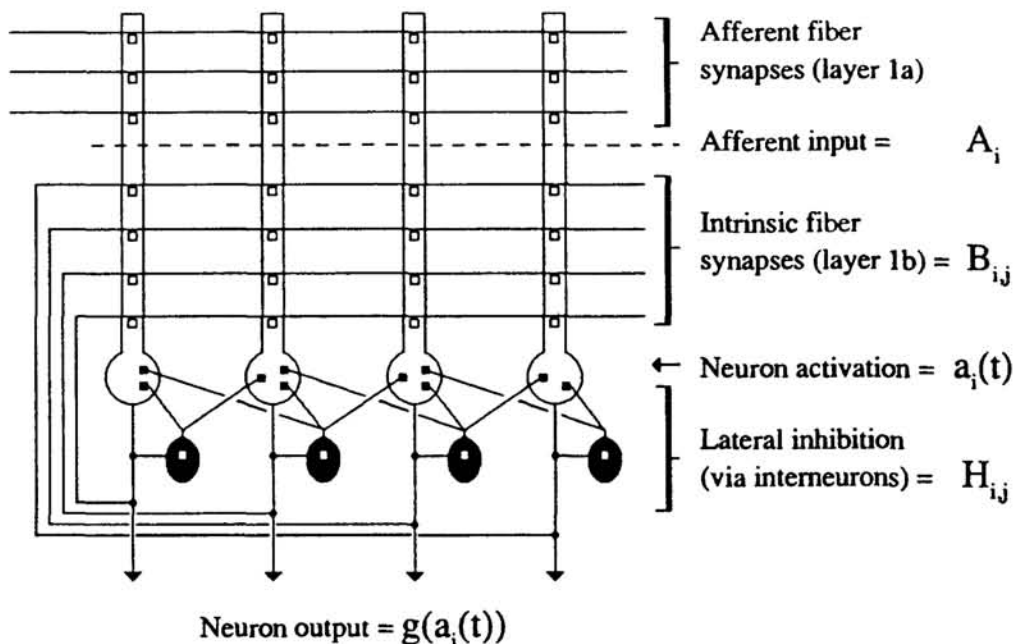

Figure 1: Schematic representation of piriform cortex, showing afferent input $A_i$ (layer 1a) and intrinsic connections $B_{ij}$ (layer 1b)

## 2   EXPERIMENTS

To study differences in the effect of acetylcholine on afferent and intrinsic fiber systems, we applied the pharmacological agent carbachol (a chemical analogue of acetylcholine) to a brain slice preparation of piriform cortex while monitoring changes in the strength of synaptic transmission associated with each fiber system. In these experiments, both extracellular and intracellular recordings demonstrated clear differences in the effects of carbachol on synaptic transmission (Hasselmo and Bower, 1991). The results in figure 2 show that synaptic potentials evoked by activating intrinsic fibers in layer 1b were strongly suppressed in the presence of

$100\mu$M carbachol, while at the same concentration, synaptic potentials evoked by stimulation of afferent fibers in layer 1a showed almost no change.

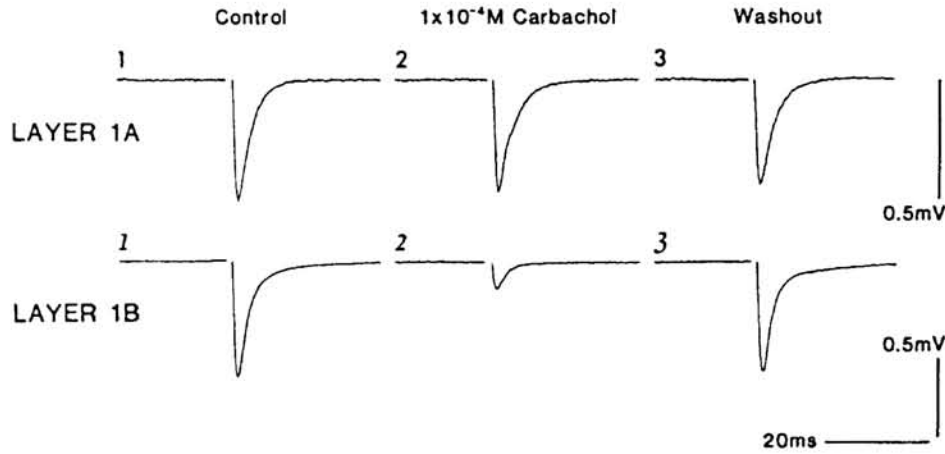

Figure 2: Synaptic potentials recorded in layer 1a and layer 1b before, during, and after perfusion with the acetylcholine analogue carbachol. Carbachol selectively suppresses layer 1b (intrinsic fiber) synaptic transmission.

These experiments demonstrate that there is a substantial difference in the neuro-chemical modulation of synapses associated with the afferent and intrinsic fiber systems within piriform cortex. Cholinergic agents selectively suppress intrinsic fiber synaptic transmission without affecting afferent fiber synaptic transmission. While interesting in purely pharmacological terms, these differential effects are even more intriguing when considered in the context of our computational models of memory function in this region.

## 3  MODELING

To investigate the effects of cholinergic suppression of intrinsic fiber synaptic transmission on associative memory function, we developed a simplified model of the piriform cortex. This simplified model is shown schematically in figure 1. At each time step, a neuron was picked at random, and its activation was updated as

$$a_i(t+1) = A_i(t) + \sum_{j=1}^{N}[(1-c)B_{ij} - H_{ij}]g(a_j(t)).$$

where $N$ = the number of neurons; $t$ = time $\in \{0,1,2,\ldots\}$; $c$ = a parameter representing the amount of acetylcholine present. $c \in [0,1]$; $a_i$ = the activation or membrane potential of neuron $i$; $g(a_i)$ = the output or firing frequency of neuron $i$ given $a_i$; $A_i$ = the input to neuron $i$, representing the afferent input from the olfactory bulb; $B_{ij}$ = the weight matrix or the synaptic strength from neuron $j$ to neuron $i$; and $H_{ij}$ = the inhibition matrix or the amount that neuron $j$ inhibits neuron $i$. To account for the local nature of inhibition in the piriform cortex, $H_{ij} = 0$

for $|i - j| > r$ and $H_{ij} = h$ for $|i - j| \leq r$, where $r$ is the inhibition radius. The function $g(a_i)$ was set to 0 if $a_i < \theta_a$, where $\theta_a = $ a firing threshold; otherwise, it was set to $\gamma_a \tanh(a_i - \theta_a)$, where $\gamma_a = $ a firing gain.

The weights were updated every $N$ time steps according to the following hebbian learning rule.

$$B_{ij} = f(W_{ij})$$
$$\Delta W_{ij} = W_{ij}(t + N) - W_{ij}(t) = (1 - c)\gamma_\ell(a_i - \theta_\ell)g(a_j)$$

The function $f(\cdot)$ is a saturating function, similar to $g(\cdot)$, used so that the weights could not become negative or grow arbitrarily large (representing a restriction on how effective synapses could become). $\gamma_\ell$ is a parameter that adjusts learning speed, and $\theta_\ell$ is a learning threshold. The weights were updated every $N$ time steps to account for the different time scales between synapse modification and neuron settling.

## 3.1   TRAINING OF THE MODEL

During learning, the model was presented with various vectors (taken to represent odors) at the input ($A_i(t)$). The network was then allowed to run and the weights to adapt.

The procedure for creating the set of vectors $\{A^m \mid m \in \{1, \ldots, M\}\}$ was: set $A_i^m = \max\{0, G(\mu, \sigma)\}$, where $G = $ gaussian with average $\mu$ and standard deviation $\sigma$, and normalize the whole vector so that $\|A^m\|^2 = N(\sigma^2 + \mu^2)$. $M = $ number of memories or odors presented to network during training, and $A_i^m = $ the input to neuron $i$ while odor $m$ is present.

During learning, in the asynchronous update equation, $A_i(t) = A_i^1$ for $\tau$ time steps, then $A_i(t) = A_i^2$ for the next $\tau$ time steps, and so on; i.e., the various odors were presented cyclically.

## 3.2   PERFORMANCE MEASURE FOR THE MODEL

The piriform cortex gets inputs from the olfactory bulb and sends outputs to other areas of the brain. Assuming that during recall the network receives noisy versions of the learned input patterns (or odors), we presume the piriform cortex performs a useful service if it reduces the chance of error in deciding which odor is present at the input. One way to quantify this is by using the minimum probability of classification error, $P_e$ (from the field of pattern recognition [2]).

For the case of 2 odors corrupted by gaussian noise, $P_e = $ the area underneath the intersection of the gaussians. For spherically symmetric gaussians with mean vectors $\mu_1$ and $\mu_2$ and identical standard deviations $\sigma$,

$$P_e = \frac{2}{\sqrt{\pi}} \int_{\frac{1}{2\sqrt{2}} \frac{d}{\sigma}}^{\infty} e^{-u^2} du$$

where $d = \|\mu_1 - \mu_2\|$. Thus, the important parameter is the amount of overlap as quantified by $d/\sigma$ — the larger the $d/\sigma$, the lower the overlap and $P_e$.

For more than 2 odors and for non-gaussian noise or non-spherically-symmetric gaussian noise, the equation for $P_e$ becomes less tractable. But keeping with the above calculations, an analogue of $d/\sigma$ was developed as follows. $\sigma_i^2$ was set $= \langle \|x - \mu_i\|^2 \rangle$, and then $\beta$ was defined as

$$\beta \equiv \sum_{i<j} \frac{\|\mu_i - \mu_j\|}{\frac{1}{2}(\sigma_i + \sigma_j)}$$

where $i, j \in \{1, \ldots, M\}$. Here, $\beta$ is the analogue of $d/\sigma$ in the previous paragraph and is similar to an average over all odor pairs of $d/\sigma$.

For the model, if $\beta$ is larger for the output vectors than for the input vectors, there is less overlap in the outputs, classification of the outputs is easier than classification of the inputs, and the model is serving a useful purpose. Thus, we use $\rho = \beta_{out}/\beta_{in}$ as the performance measure.

## 3.3   TESTING THE MODEL

The model was designed to show whether the presence of acetylcholine has any influence on learning performance. To that end, the model was allowed to learn for a time with various levels of acetylcholine present, and then acetylcholine was turned off and the model was tested.

For testing, weight adaptation was turned off, acetylcholine influence was turned off ($c = 0$), noisy versions of the various odors presented during learning were presented at the input, and the network was allowed to settle. From these noisy input/output pairs, $\sigma$'s could be estimated, $\beta_{in}$ and $\beta_{out}$ could be calculated, and finally $\rho$ could be calculated. Then, the state of the network could either be reset (for a new learning run) or be set to what it was before the test (so that learning could continue as if uninterrupted).

## 3.4   RESULTS OF TESTING

A typical example of a test run is shown in figure 3. There, $c$ was varied from 0 (no acetylcholine) to .9 (a large concentration), and the various other parameters were: $N = 10$, $M = 10$, $r = 2$, $h = .3$, $\gamma_a = 1$, $\theta_a = 1$, $\gamma_\ell = 10^{-3}$, $\theta_\ell = 1$, and $\tau = 10$. In the figure, large dark rectangles represent larger values of $\rho$. Small or non-existant rectangles represent values of $\rho \leq 1$.

Notice that, for a fixed amount of acetylcholine, the model's performance rises and then falls over time. Ideally, the performance should rise and then flatten out, as further learning should not degrade performance. The weight adaptation equation used in the model was not optimized to preclude overlearning (where all of the weights being reinforced have saturated to the largest allowed value). In principle, the function $f(\cdot)$ could be used for this, perhaps in conjunction with a weight decay term. This was not of great concern since the peak performance is what indicates whether or not acetylcholine has a useful effect. Also, the more acetylcholine present, the longer the learning took. This is reasonable as, before saturation, $\Delta W \propto (1-c)$.

Figure 4 shows maximum average performance for various values of acetylcholine. Averages were calculated by doing many tests like the one above. This is useful as

the odor inputs and the individual tests are stochastic in nature. Obviously, the larger values of acetylcholine enhance performance.

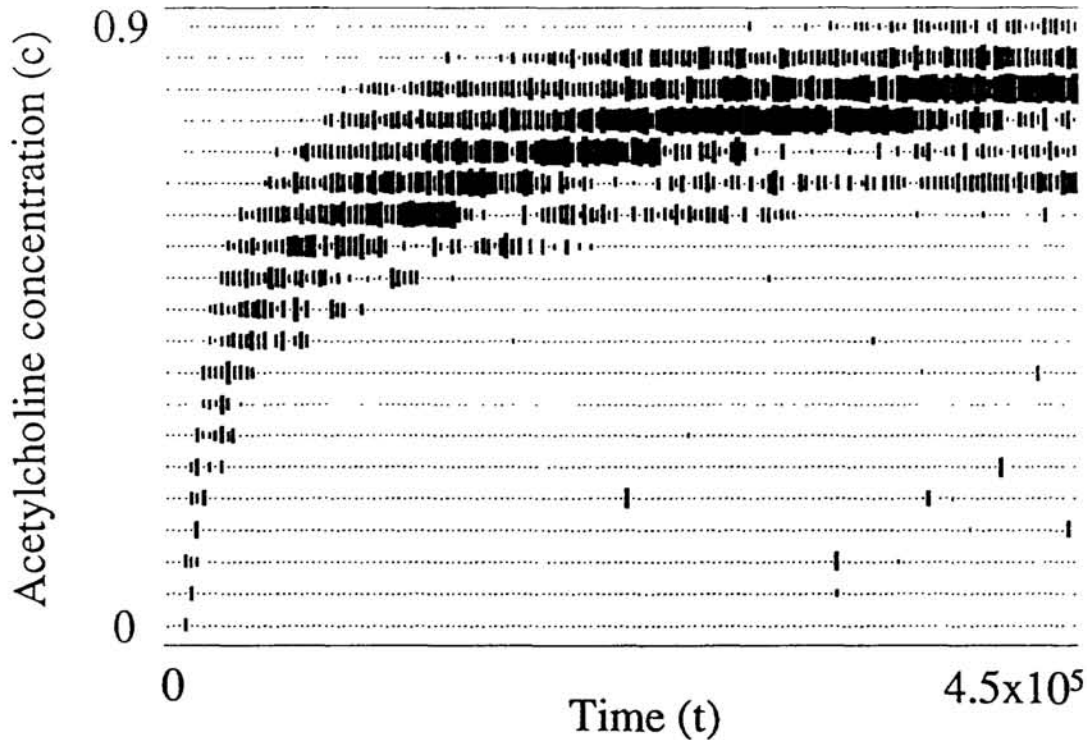

Figure 3: Sample test run, with time on the horizontal axis and acetylcholine level on the vertical axis. Larger black rectangles indicate better performance.

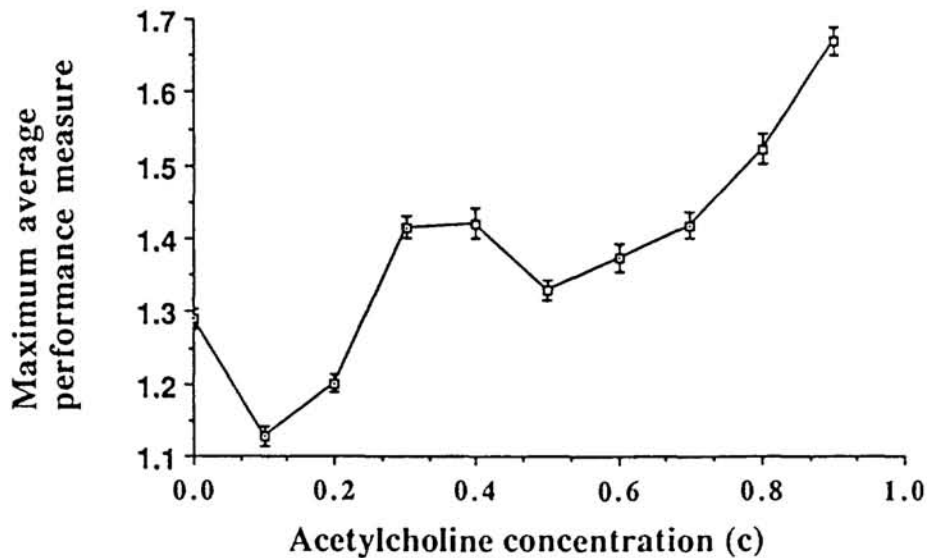

Figure 4: Maximum average performance vs. acetylcholine level. Acetylcholine increases the performance level attained.

## 4   CONCLUSION

The results from the model show that suppression of connections between cells within the piriform cortex during learning enhances the performance during recall. Thus, acetylcholine released in the cortex during learning may enhance associative memory function. These results may explain some of the behavioral evidence for the role of acetylcholine in memory function and predict that acetylcholine may be released in cortical structures preferentially during learning. Further biological experiments are necessary to confirm this prediction.

**Acknowledgements**

This work was supported by ONR contracts N00014-88-K-0513 and N00014-87-K-0377 and NIH postdoctoral training grant NS07251.

**References**

J.A. Anderson, J.W. Silverstein, S.A. Ritz and R.S. Jones (1977) Distinctive features, categorical perception, and probability learning: Some applications of a neural model. *Psychol. Rev.* 84: 413-451.

J.M. Bower (1990) Reverse engineering the nervous system: An anatomical, physiological and computer based approach. In S. Zornetzer, J. Davis and C. Lau (eds.), *An Introduction to Neural and Electronic Networks.* San Diego: Academic Press.

R. Duda and P. Hart (1973), *Pattern Classification and Scene Analysis*, New York: Wiley.

L.B. Haberly and J.M. Bower (1989) Olfactory cortex: Model circuit for study of associative memory? *Trends Neurosci.* 12: 258-264.

J.J. Hagan and R.G.M. Morris (1989) The cholinergic hypothesis of memory: A review of animal experiments. In L.L. Iversen, S.D. Iversen and S.H. Snyder (eds.) *Handbook of Psychopharmacology* Vol. 20 New York: Plenum Press.

M.E. Hasselmo, M.A. Wilson, B.P. Anderson and J.M. Bower (1991) Associative memory function in piriform (olfactory) cortex: Computational modeling and neuropharmacology. In: *Cold Spring Harbor Symposium on Quantitative Biology: The Brain.* Cold Spring Harbor: Cold Spring Harbor Laboratory.

M.E. Hasselmo and J.M. Bower (1991) Cholinergic suppression specific to intrinsic not afferent fiber synapses in piriform (olfactory) cortex. *J. Neurophysiol.* in press.

T. Kohonen, P. Lehtio, J. Rovamo, J. Hyvarinen, K. Bry and L. Vainio (1977) A principle of neural associative memory. *Neurosci.* 2:1065-1076.

M.D. Kopelman (1986) The cholinergic neurotransmitter system in human memory and dementia: A review. *Quart. J. Exp. Psychol.* 38A:535-573.

M.A. Wilson and J.M. Bower (1988) A computer simulation of olfactory cortex with functional implications for storage and retrieval of olfactory information. In D. Anderson (ed.) *Neural Information Processing Systems.* AIP Press: New York.

## Footnotes

[1] For descriptions of standard associative memory models, see for example (Anderson *et al.*, 1977; Kohonen *et al.*, 1977).

[2]See, for example, (Duda and Hart, 1973).
